# Exploiting generative models in discriminative classifiers

**Tommi S. Jaakkola***
MIT Artificial Intelligence Laboratorio
545 Technology Square
Cambridge, MA 02139

**David Haussler**
Department of Computer Science
University of California
Santa Cruz, CA 95064

## Abstract

Generative probability models such as hidden Markov models provide a principled way of treating missing information and dealing with variable length sequences. On the other hand, discriminative methods such as support vector machines enable us to construct flexible decision boundaries and often result in classification performance superior to that of the model based approaches. An ideal classifier should combine these two complementary approaches. In this paper, we develop a natural way of achieving this combination by deriving kernel functions for use in discriminative methods such as support vector machines from generative probability models. We provide a theoretical justification for this combination as well as demonstrate a substantial improvement in the classification performance in the context of DNA and protein sequence analysis.

## 1  Introduction

Speech, vision, text and biosequence data can be difficult to deal with in the context of simple statistical classification problems. Because the examples to be classified are often sequences or arrays of variable size that may have been distorted in particular ways, it is common to estimate a generative model for such data, and then use Bayes rule to obtain a classifier from this model. However, many discriminative methods, which directly estimate a posterior probability for a class label (as in Gaussian process classifiers [5]) or a discriminant function for the class label (as in support vector machines [6]) have in other areas proven to be superior to

---

generative models for classification problems. The problem is that there has been no systematic way to extract features or metric relations between examples for use with discriminative methods in the context of difficult data types such as those listed above. Here we propose a general method for extracting these discriminatory features using a generative model. While the features we propose are generally applicable, they are most naturally suited to kernel methods.

## 2    Kernel methods

Here we provide a brief introduction to kernel methods; see, e.g., [6] [5] for more details. Suppose now that we have a training set of examples $X_i$ and corresponding binary labels $S_i$ ($\pm1$). In kernel methods, as we define them, the label for a new example $X$ is obtained from a weighted sum of the training labels. The weighting of each training label $S_i$ consists of two parts: 1) the overall importance of the example $X_i$ as summarized with a coefficient $\lambda$, and 2) a measure of pairwise "similarity" between between $X_i$ and $X$, expressed in terms of a kernel function $K(X_i, X)$. The predicted label $\hat{S}$ for the new example $X$ is derived from the following rule:

$$\hat{S} = \text{sign}\left( \sum_i S_i \lambda_i K(X_i, X) \right) \tag{1}$$

We note that this class of kernel methods also includes probabilistic classifiers, in which case the above rule refers to the label with the maximum probability. The free parameters in the classification rule are the coefficients $\lambda_i$ and to some degree also the kernel function $K$. To pin down a particular kernel method, two things need to be clarified. First, we must define a classification loss, or equivalently, the optimization problem to solve to determine appropriate values for the coefficients $\lambda_i$. Slight variations in the optimization problem can take us from support vector machines to generalized linear models. The second and the more important issue is the choice of the kernel function – the main topic of this paper. We begin with a brief illustration of generalized linear models as kernel methods.

### 2.1    Generalized linear models

For concreteness we consider here only logistic regression models, while emphasizing that the ideas are applicable to a larger class of models[1]. In logistic regression models, the probability of the label $S$ given the example $X$ and a parameter vector $\theta$ is given by[2]

$$P(S|X, \theta) = \sigma\left( S\theta^T X \right) \tag{2}$$

where $\sigma(z) = (1 + e^{-z})^{-1}$ is the logistic function. To control the complexity of the model when the number of training examples is small we can assign a prior distribution $P(\theta)$ over the parameters. We assume here that the prior is a zero mean Gaussian with a possibly full covariance matrix $\Sigma$. The maximum *a posteriori* (MAP) estimate for the parameters $\theta$ given a training set of examples is found by

maximizing the following penalized log-likelihood:

$$\sum_\imath \log P(S_\imath | X_\imath, \theta) + \log P(\theta) \ = \ \sum_\imath \log \sigma\left(S_\imath \theta^T X_\imath\right) - \frac{1}{2}\theta^T \Sigma^{-1}\theta + c \quad (3)$$

where the constant $c$ does not depend on $\theta$. It is straightforward to show. simply by taking the gradient with respect to the parameters, that the solution to this (concave) maximization problem can be written as[3]

$$\hat{\theta} \ = \ \sum_\imath S_\imath \lambda_\imath \Sigma X_\imath. \quad \text{where} \quad \lambda_\imath \ = \ \frac{\partial}{\partial z} \log \sigma(z)\Big|_{z = S_\imath \hat{\theta}^T X_\imath} \quad (4)$$

Note that the coefficients $\lambda_\imath$ appear as weights on the training examples as in the definition of the kernel methods. Indeed. inserting the above solution back into the conditional probability model gives

$$P(S|X. \hat{\theta}) = \sigma\left( S \sum_\imath S_\imath \lambda_\imath \left(X_\imath^T \Sigma X\right) \right) \quad (5)$$

By identifying $K(X_\imath. X) = X_\imath^T \Sigma X$ and noting that the label with the maximum probability is the one that has the same sign as the sum in the argument. this gives the decision rule (1).

Through the above derivation. we have written the primal parameters $\theta$ in terms of the dual coefficients $\lambda_\imath$[4]. Consequently. the penalized log-likelihood function can be also written entirely in terms of $\lambda_\imath$: the resulting likelihood function specifies how the coefficients are to be optimized. This optimization problem has a unique solution and can be put into a generic form. Also, the form of the kernel function that establishes the connection between the logistic regression model and a kernel classifier is rather specific, i.e.. has the inner product form $K(X_\imath. X) = X_\imath^T \Sigma X$. However. as long as the examples here can be replaced with feature vectors derived from the examples. this form of the kernel function is the most general. We discuss this further in the next section.

## 3 The kernel function

For a general kernel function to be valid. roughly speaking it only needs to be positive semi-definite (see e.g. [7]). According to the Mercer's theorem. any such valid kernel function admits a representation as a simple inner product between suitably defined feature vectors. i.e.. $K(X_\imath. X_\jmath) = \phi_{X_\imath}^T \phi_{X_\jmath}$. where the feature vectors come from some fixed mapping $X \to \phi_X$. For example. in the previous section the kernel function had the form $X_\imath^T \Sigma X_\jmath$, which is a simple inner product for the transformed feature vector $\phi_X = \Sigma^{\frac{1}{2}} X$.

Specifying a simple inner product in the feature space defines a Euclidean metric space. Consequently. the Euclidean distances between the feature vectors are obtained directly from the kernel function: with the shorthand notation $K_{\imath\jmath} =$

$K(X_i, X_j)$ we get $\|\phi_{X_i} - \phi_{X_j}\|^2 = K_{ii} - 2K_{ij} + K_{jj}$. In addition to defining the metric structure in the feature space, the kernel defines a pseudo metric in the original example space through $D(X_i, X_j) = \|\phi_{X_i} - \phi_{X_j}\|$. Thus the kernel embodies prior assumptions about the metric relations between the original examples. No systematic procedure has been proposed for finding kernel functions, let alone finding ones that naturally handle variable length examples etc. This is the topic of the next section.

## 4  Kernels from generative probability models: the Fisher kernel

The key idea here is to derive the kernel function from a generative probability model. We arrive at the same kernel function from two different perspectives, that of enhancing the discriminative power of the model and from an attempt to find a natural comparison between examples induced by the generative model. Both of these ideas are developed in more detail in the longer version of this paper[4].

We have seen in the previous section that defining the kernel function automatically implies assumptions about metric relations between the examples. We argue that these metric relations should be defined directly from a generative probability model $P(X|\theta)$. To capture the generative process in a metric between examples we use the gradient space of the generative model. The gradient of the log-likelihood with respect to a parameter describes how that parameter contributes to the process of generating a particular example[5]. This gradient space also naturally preserves all the structural assumptions that the model encodes about the generation process.

To develop this idea more generally, consider a parametric class of models $P(X|\theta)$, $\theta \in \Theta$. This class of probability models defines a Riemannian manifold $M_\Theta$ with a local metric given by the Fisher information matrix[6] $I$, where $I = E_X\{U_X U_X^T\}$, $U_X = \nabla_\theta \log P(X|\theta)$, and the expectation is over $P(X|\theta)$ (see e.g. [1]). The gradient of the log-likelihood, $U_X$, is called the *Fisher score*, and plays a fundamental role in our development. The local metric on $M_\Theta$ defines a distance between the current model $P(X|\theta)$ and a nearby model $P(X|\theta+\delta)$. This distance is given by $D(\theta, \theta+\delta) = \frac{1}{2}\delta^T I \delta$, which also approximates the KL-divergence between the two models for a sufficiently small $\delta$.

The Fisher score $U_X = \nabla_\theta \log P(X|\theta)$ maps an example $X$ into a feature vector that is a point in the gradient space of the manifold $M_\Theta$. We call this the *Fisher score mapping*. This gradient $U_X$ can be used to define the direction of steepest ascent in $\log P(X|\theta)$ for the example $X$ *along the manifold*, i.e., the gradient in the direction $\delta$ that maximizes $\log P(X|\theta)$ while traversing the minimum distance in the manifold as defined by $D(\theta, \theta + \delta)$. This latter gradient is known as the natural gradient (see e.g. [1]) and is obtained from the ordinary gradient via $\phi_X = I^{-1}U_X$. We will call the mapping $X \to \phi_X$ the *natural mapping* of examples into feature vectors[7]. The natural kernel of this mapping is the inner product between these

feature vectors relative to the local Riemannian metric:

$$K(X_i, X_j) \propto \phi_{X_i}^T I \phi_{X_j} = U_{X_i}^T I^{-1} U_{X_j} \tag{6}$$

We call this the Fisher kernel owing to the fundamental role played by the Fisher scores in its definition. The role of the information matrix is less significant; indeed, in the context of logistic regression models, the matrix appearing in the middle of the feature vectors relates to the covariance matrix of a Gaussian prior, as show above. Thus, asymptotically, the information matrix is immaterial, and the simpler kernel $K_U(X_i, X_j) \propto U_{X_i}^T U_{X_j}$ provides a suitable substitute for the Fisher kernel.

We emphasize that the Fisher kernel defined above provides only the basic comparison between the examples, defining what is meant by an "inner product" between the examples when the examples are objects of various types (e.g. variable length sequences). The way such a kernel function is used in a discriminative classifier is not specified here. Using the Fisher kernel directly in a kernel classifier, for example, amounts to finding a linear separating hyper-plane in the natural gradient (or Fisher score) feature space. The examples may not be linearly separable in this feature space even though the natural metric structure is given by the Fisher kernel. It may be advantageous to search in the space of quadratic (or higher order) decision boundaries, which is equivalent to transforming the Fisher kernel according to $\tilde{K}(X_i, X_j) = (1 + K(X_i, X_j))^m$ and using the resulting kernel $\tilde{K}$ in the classifier.

We are now ready to state a few properties of the Fisher kernel function. So long as the probability model $P(X|\theta)$ is suitably regular then the Fisher kernel derived from it is a) a valid kernel function and b) invariant to any invertible (and differentiable) transformation of the model parameters. The rather informally stated theorem below motivates the use of this kernel function in a classification setting.

**Theorem 1** *Given any suitably regular probability model $P(X|\theta)$ with parameters $\theta$ and assuming that the classification label is included as a latent variable, the Fisher kernel $K(X_i, X_j) = U_{X_i}^T I^{-1} U_{X_j}$ derived from this model and employed in a kernel classifier is, asymptotically, never inferior to the MAP decision rule from this model.*

The proofs and other related theorems are presented in the longer version of this paper [4].

To summarize, we have defined a generic procedure for obtaining kernel functions from generative probability models. Consequently the benefits of generative models are immediately available to the discriminative classifier employing this kernel function. We now turn the experimental demonstration of the effectiveness of such a combined classifier.

## 5   Experimental results

Here we consider two relevant examples from biosequence analysis and compare the performance of the combined classifier to the best generative models used in these problems. We start with a DNA splice site classification problem, where the objective is to recognize true splice sites, i.e., the boundaries between expressed regions (exons) in a gene and the intermediate regions (introns). The data set used in our experiments consisted of 9350 DNA fragments from *C. elegans*. Each of the

2029 true examples is a sequence $X$ over the DNA alphabet $\{A, G, T, C\}$ of length 25; the 7321 false examples are similar sequences that occur near but not at 5' splice sites. All recognition rates we report on this data set are averages from 7-fold cross-validation.

To use the combined classifier in this setting requires us to choose a generative model for the purpose of deriving the kernel function. In order to test how much the performance of the combined classifier depends on the quality of the underlying generative model, we chose the poorest model possible. This is the model where the DNA residue in each position in the fragment is chosen independently of others, i.e., $P(X|\theta) = \prod_{i=1}^{25} P(X_i|\theta_i)$ and, furthermore, the parameters $\theta_i$ are set such that $P(X_i|\theta_i) = 1/4$ for all $i$ and all $X_i \in \{A, G, T, C\}$. This model assigns the same probability to all examples $X$. We can still derive the Fisher kernel from such a model and use it in a discriminative classifier. In this case we used a logistic regression model as in (5) with a quadratic Fisher kernel $\tilde{K}(X_i, X_j) = (1 + K(X_i, X_j))^2$. Figure 1 shows the recognition performance of this kernel method, using the poor generative model, in comparison to the recognition performance of a naive Bayes model or a hierarchical mixture model. The comparison is summarized in ROC style curves plotting false positive errors (the errors of accepting false examples) as a function of false negative errors (the errors of missing true examples) when we vary the classification bias for the labels. The curves show that even with such a poor underlying generative model, the combined classifier is consistently better than either of the better generative models alone.

In the second and more serious application of the combined classifier, we consider the well-known problem of recognizing remote homologies (evolutionary/structural similarities) between protein sequences[8] that have low residue identity. Considerable recent work has been done in refining hidden Markov models for this purpose as reviewed in [2], and such models current achieve the best performance. We use these state-of-the-art HMMs as comparison cases and also as sources for deriving the kernel function. Here we used logistic regression with the simple kernel $K_U(X_i, X_j)$, as the number of parameters in the HMMs was several thousand.

The experiment was set up as follows. We picked a particular superfamily (glycosyl-transferases) from the TIM-barrel fold in the SCOP protein structure classification [3], and left out one of the four major families in this superfamily for testing while training the HMM as well as the combined classifier on sequences corresponding to the remaining three families. The false training examples for the discriminative method came from those sequences in the same fold but not in the same superfamily. The test sequences consisted of the left-out family (true examples) and proteins outside the TIM barrel fold (false examples). The number of training examples varied around 100 depending on the left-out family. As the sequences among the four glycosyltransferase families are extremely different, this is a challenging discrimination problem. Figure 1c shows the recognition performance curves for the HMM and the corresponding kernel method, averaged over the four-way cross validation. The combined classifier yields a substantial improvement in performance over the HMM alone.

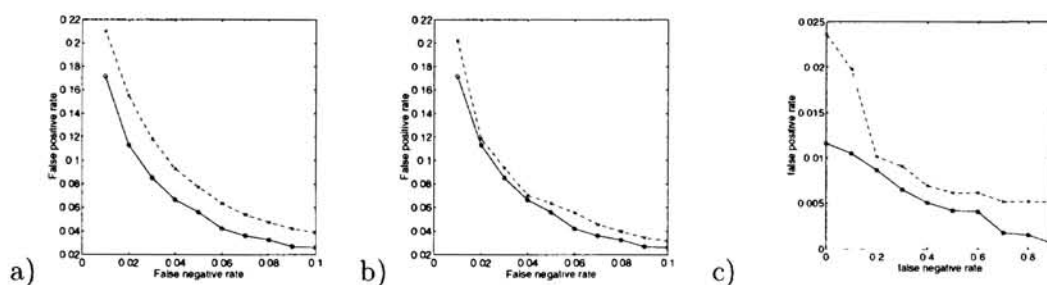

Figure 1: a) & b) Comparison of classification performance between a kernel classifiers from the uniform model (solid line) and a mixture model (dashed line). In a) the mixture model is a naive Bayes model and in b) it has three components in each class. c) Comparison of homology recognition performance between a hidden Markov model (dashed line) and the corresponding kernel classifier (solid line).

## 6 Discussion

The model based kernel function derived in this paper provides a generic mechanism for incorporating generative models into discriminative classifiers. For discrimination, the resulting combined classifier is guaranteed to be superior to the generative model alone with little additional computational cost. We note that the power of the new classifier arises to a large extent from the use of Fisher scores as features in place of original examples. It is possible to use these features with any classifier, e.g. a feed-forward neural net, but kernel methods are most naturally suited for incorporating them.

Finally we note that while we have used classification to guide the development of the kernel function, the results are directly applicable to regression, clustering, or even interpolation problems, all of which can easily exploit metric relations among the examples defined by the Fisher kernel.

## Footnotes

[1]Specifically, it applies to all generalized linear models whose transfer functions are log-concave.

[2]Here we assume that the constant $+1$ is appended to every feature vector $X$ so that an adjustable bias term is included in the inner product $\theta^T X$.

[3]This corresponds to a Legendre transformation of the loss functions $\log \sigma(z)$.

[4]This is possible for all those $\theta$ that could arise as solutions to the maximum penalized likelihood problem: in other words, for all relevant $\theta$.

[5]For the exponential family of distributions, under the natural parameterization $\theta$, these gradients, less a normalization constant that depends on $\theta$, form sufficient statistics for the example.

[6]For simplicity we have suppressed the dependence of $I$ and $U_X$ on the parameter setting $\theta$, or equivalently, on the position in the manifold.

[7]Again, we have suppressed dependence on the parameter setting $\theta$ here.

[8]These are variable length sequences thus rendering many discriminative methods inapplicable.

## References

[1] S.-I. Amari. Natural gradient works efficiently in learning. *Neural Computation*, 10:251–276, 1998.

[2] R. Durbin, S. Eddy, A. Krogh, and G. Mitchison. *Biological Sequence Analysis: Probabilistic Models of Proteins and Nucleic Acids.* Cambridge University Press, 1998.

[3] T. Hubbard, A. Murzin, S. Brenner, and C. Chothia. SCOP: a structural classification of proteins database. *NAR*, 25(1):236–9, Jan. 1997.

[4] T. S. Jaakkola and D. Haussler. Exploiting generative models in discriminative classifiers. 1998. Revised and extended version. Will be available from http://www.ai.mit.edu/~tommi.

[5] D. J. C. MacKay. Introduction to gaussian processes. 1997. Available from http://wol.ra.phy.cam.ac.uk/mackay/.

[6] V. Vapnik. *The nature of statistical learning theory.* Springer-Verlag. 1995.

[7] G. Wahba. *Spline models for observational data.* CBMS-NSF Regional Conference Series in Applied Mathematics, 1990.